# Unsupervised and supervised clustering: the mutual information between parameters and observations

Didier Herschkowitz        Jean-Pierre Nadal

Laboratoire de Physique Statistique de l'E.N.S.*
Ecole Normale Supérieure
24, rue Lhomond - 75231 Paris cedex 05, France
herschko@lps.ens.fr        nadal@lps.ens.fr
http://www.lps.ens.fr/~risc/rescomp

## Abstract

Recent works in parameter estimation and neural coding have demonstrated that optimal performance are related to the mutual information between parameters and data. We consider the mutual information in the case where the dependency in the parameter (a vector $\theta$) of the conditional p.d.f. of each observation (a vector $\xi$), is through the scalar product $\theta.\xi$ only. We derive bounds and asymptotic behaviour for the mutual information and compare with results obtained on the same model with the "replica technique".

## 1   INTRODUCTION

In this contribution we consider an unsupervised clustering task. Recent results on neural coding and parameter estimation (supervised and unsupervised learning tasks) show that the mutual information between data and parameters (equivalently between neural activities and stimulus) is a relevant tool for deriving optimal performances (Clarke and Barron, 1990; Nadal and Parga, 1994; Opper and Kinzel, 1995; Haussler and Opper, 1995; Opper and Haussler, 1995; Rissanen, 1996; Brunel and Nadal 1998).

With this tool we analyze a particular case which has been studied extensively with the "replica technique" in the framework of statistical mechanics (Watkin and Nadal, 1994; Reimann and Van den Broeck, 1996; Buhot and Gordon, 1998). After introducing the model in the next section, we consider the mutual information between the patterns and the parameter. We derive a bound on it which is of interest for not too large $p$. We show how the "free energy" associated to *Gibbs learning* is related to the mutual information. We then compare the exact results with replica calculations. We show that the asymptotic behaviour ($p >> N$) of the mutual information is in agreement with the exact result which is known to be related to the *Fisher information* (Clarke and Barron, 1990; Rissanen, 1996; Brunel and Nadal 1998). However for moderate values of $\alpha = p/N$, we can eliminate false solutions of the replica calculation. Finally, we give bounds related to the mutual information between the parameter and its estimators, and discuss common features of parameter estimation and neural coding.

## 2 THE MODEL

We consider the problem where a direction $\theta$ (a unit vector) of dimension $N$ has to be found based on the observation of $p$ patterns. The probability distribution of the patterns is uniform except in the unknown symmetry-breaking direction $\theta$. Various instances of this problem have been studied recently within the satistical mechanics framework, making use of the *replica technique* (Watkin and Nadal, 1994; Reimann and Van den Broeck, 1996; Buhot and Gordon, 1998). More specifically it is assumed that a set of patterns $D = \{\xi^\mu\}_{\mu=1}^p$ is generated by $p$ independant samplings from a non-uniform probability distribution $P(\xi|\theta)$ where $\theta = \{\theta_1, ..., \theta_N\}$ represents the symmetry-breaking orientation. The probability is written in the form:

$$P(\xi|\theta) = \frac{1}{\sqrt[N]{2\pi}} exp(-\frac{\xi^2}{2} - V(\lambda)) \tag{1}$$

where $N$ is the dimension of the space, $\lambda = \theta.\xi$ is the overlap and $V(\lambda)$ characterizes the structure of the data in the breaking direction. As justified within the Bayesian and Statistical Physics frameworks, one has to consider a *prior* distribution on the parameter space, $\rho(\theta)$, e.g. the uniform distribution on the sphere.

The mutual information $I(D|\theta)$ between the data and $\theta$ is defined by

$$I(D|\theta) = \int d\theta dD P(\theta) P(D|\theta) ln(\frac{P(D|\theta)}{P(D)}) \tag{2}$$

It can be rewritten:

$$\frac{I(D|\theta)}{N} = -\alpha <V(\lambda)> - \frac{<<ln(Z)>>}{N} \tag{3}$$

where

$$Z = \int_{-\infty}^{\infty} d\theta \rho(\theta) exp(-\sum_{\mu=1}^p V(\lambda^\mu)) \tag{4}$$

In the statistical physics literature $-\ln Z$ is a "free energy". The brackets $<< .. >>$ stand for the average over the pattern distribution, and $< .. >$ is the average over the resulting overlap distribution. We will consider properties valid for any $N$ and any $p$, others for $p >> N$, and the replica calculations are valid for $N$ and $p$ large at any given value of $\alpha = \frac{p}{N}$.

## 3  LINEAR BOUND

The mutual information, a positive quantity, cannot grow faster than linearly in the amount of data, $p$. We derive the simple linear bound:

$$I(D|\theta) \leq -p < V(\lambda) > \tag{5}$$

We proove the inequality for the case $< \lambda >= 0$. The extension to the case $< \lambda > \neq 0$ is straightforward. The mutual information can be written as $I = H(D) - H(D|\theta)$. The calculation of $H(D|\theta)$ is straightforward:

$$H(D|\theta) = \frac{pN}{2}ln(e2\pi) + \frac{p}{2}(< \lambda^2 > -1) + p < V > \tag{6}$$

Now, the entropy of the data $H(D) = -\int dD P(D) ln P(D)$ is lower or equal to the entropy of a Gaussian distribution with the same variance. We thus calculate the covariance matrix of the data

$$<< \xi_i^\mu \xi_j^\nu >> = \delta_{\mu\nu}( \delta_{ij} + (< \lambda^2 > -1)\overline{\theta_i\theta_j}) \tag{7}$$

where $\overline{(.)}$ denotes the average over the parameter distribution. We then have

$$H(D) \leq \frac{pN}{2}ln(2\pi e) + \frac{p}{2}\sum_{i=1}^{N} ln(1 + (< \lambda^2 > -1)\gamma_i) \tag{8}$$

where $\gamma_i$ are the eigen value of the matrix $\overline{\theta_i\theta_j}$. Using $\sum_{i=1}^{N} \overline{\theta_i^2} = 1$ and the property $ln(1 + x) \leq x$ we obtain

$$H(D) \leq \frac{pN}{2}ln(2\pi e) + \frac{p}{2}(< \lambda^2 > -1) \tag{9}$$

Putting (9) and (6) together, we find the inequality (5). ¿From this and (3) it follows also

$$p < V > \leq - << ln(Z) >> \leq 0 \tag{10}$$

## 4  REPLICA CALCULATIONS

In the limit $N \to \infty$ with $\alpha$ finite, the free energy becomes self-averaging, that is equal to its average, and its calculation can be performed by standard replica technique. This calculation is the same as calculations related to Gibbs learning, done in (Reimann and van den Broeck, 1996, Buhot and Gordon, 1998), but the interpretation of the order parameters is different. Assuming replica symmetry, we reproduce in fig.2 results from (Buhot and Gordon, 1998) for the behaviour with $\alpha$ of $Q$ which is the typical overlap between two directions compatible with the data. The overlap distribution $P(\lambda)$ was chosen to get patterns distributed according to two clusters along the symmetry-breaking direction

$$P(\lambda) = \frac{1}{2\sigma\sqrt{2\pi}} \sum_{\epsilon=\pm 1} exp(-\frac{(\lambda - \epsilon\rho)^2}{2\sigma^2}) \tag{11}$$

In fig.2 and fig.1 we show the corresponding behaviour of the average free energy and of the mutual information.

## 4.1 Discussion

Up to $\alpha_1$, $Q = 0$ and the mutual information is in a purely linear phase $\frac{I(\theta|D)}{N} = -\alpha < V(\lambda) >$. This correspond to a regime where the data have no correlations. For $\alpha \geq \alpha_1$, the replica calculation admits up to three differents solutions. In view of the fact that the mutual information can never decrease with $\alpha$ and that the average free energy can not be positive, it follows that only two behaviours are acceptable. In the first, $Q$ leaves the solution $Q = 0$ at $\alpha_1$, and follows the lower branch until $\alpha_3$ where it jumps to the upper branch. This is the stable way. The second possibility is that $Q = 0$ until $\alpha_2$ where it directly jumps to the upper branch. In (Buhot and Gordon, 1998), it has been suggested that one can reach the upper branch, well before $\alpha_3$. Here we have thus shown that it is only possible from $\alpha_2$. It remains also the possibility of a replica symetry breacking phase in this range of $\alpha$.

In the limit $\alpha \to \infty$ the replica calculus gives for the behaviour of the mutual information

$$I(D|\theta) \cong \frac{N}{2} ln(\alpha < (\frac{dV(\lambda)}{d\lambda})^2 >) \qquad (12)$$

The r.h.s can be shown to be equal to half the logarithm of the determinant of the *Fisher information* matrix, which is the exact asymptotic behaviour (Clarke and Barron, 1990; Brunel and Nadal, 1998). It can be shown that this behaviour for $p >> N$ implies that the best possible estimator based on the data will saturate the *Cramer-Rao* bound (see e.g. Blahut, 1988). It has already been noted that the asymptotics performance in estimating the direction, as computed by the replica technique, saturate this bound (Van den Broeck, 1997). What we have check here is that this manifests itself in the behaviour of the mutual information for large $\alpha$.

## 4.2 Bounds for specific estimators

Given the data $D$, one wants to find an estimate $J$ of the parameter. The amount of information $I(D|\theta)$ limits the performance of the estimator. Indeed, one has $I(J|\theta) \leq I(D|\theta)$. This basic relationship allows to derive interesting bounds based on the choice of particular estimators. We consider first *Gibbs learning*, which consists in sampling a direction $J$ from the 'a posteriori' probability $P(J|D) = P(D|J)\rho(J) / P(D)$. In this particular case, the differential entropy of the estimator $J$ and of the parameter $\theta$ are equal $H(J) = H(\theta)$. If $1 - Q_g{}^2$ is the variance of the *Gibbs* estimator one gets, for a Gaussian prior on $\theta$, the relations

$$-\frac{N}{2} ln(1 - Q_g{}^2) \leq I_{Gibbs}(J|\theta) \leq I(D|\theta) \qquad (13)$$

These relations together with the linear bound (5) allows to bound the order parameter $Q_g$ for small $\alpha$ where this bound is of interest.

The *Bayes estimator* consists in taking for $J$ the center of mass of the 'a posteriori' probability. In the limit $\alpha \to \infty$, this distribution becomes Gaussian centered at its most probable value. We can thus assume $P_{Bayes}(J|\theta)$ to be Gaussian with mean $Q_b\theta$ and variance $1 - Q_b{}^2$, then the first inequality in (13) (with $Q_g$ replaced by $Q_b$ and *Gibbs* by *Bayes*) is an equality. Then using the Cramer-Rao bound on the variance of the estimator, that is $(1 - Q_b^2)/Q_b^2 \geq (\alpha < (dV/d\lambda)^2 >)^{-1}$, one can bound the mutual information for the Bayes estimator

$$I_{Bayes}(J|\theta) \leq \frac{N}{2} ln(1 + \alpha < (\frac{dV(\lambda)}{d\lambda})^2 >) \qquad (14)$$

These different quantities are shown on fig.1.

# 5   CONCLUSION

We have studied the mutual information between data and parameter in a problem of unsupervised clustering: we derived bounds, asymptotic behaviour, and compared these results with replica calculations. Most of the results concerning the behaviour of the mutual information, observed for this particular clustering task, are "universal", in that they will be qualitatively the same for any problem which can be formulated as either a parameter estimation task or a neural coding/signal processing task. In particular, there is a linear regime for small enough amount of data (number of coding cells), up to a maximal value related to the VC dimension of the system. For large data size, the behaviour is logarithmic - that is $I \sim \ln p$ (Nadal and Parga, 1994; Opper and Haussler, 1995) or $\frac{1}{2} \ln p$ (Clarke and Barron, 1990; Opper and Haussler, 1995; Brunel and Nadal, 1998) depending on the smoothness of the model. A more detailed review with more such universal features, exact bounds and relations between unsupervised and supervised learning will be presented elsewhere. (Nadal, Herschkowitz, to appear in Phys. rev. E).

### Acknowledgements

We thank Arnaud Buhot and Mirta Gordon for stimulating discussions. This work has been partly supported by the French contract DGA 96 2557A/DSP.

## Footnotes

*Laboratory associated with C.N.R.S. (U.R.A. 1306), ENS, and Universities Paris VI and Paris VII.

# References

[B88]        R. E. Blahut, Addison-Wesley, Cambridge MA, 1998.

[BG98]       A. Buhot and M. Gordon. *Phys. Rev. E*, 57(3):3326–3333, 1998.

[BN98]       N. Brunel and J.-P. Nadal. *Neural Computation*, to appear, 1998.

[CB90]       B. S. Clarke and A. R. Barron. *IEEE Trans. on Information Theory*, 36 (3):453–471, 1990.

[HO95]       D. Haussler and M. Opper. conditionally independent observations. In *VIIIth Ann. Workshop on Computational Learning Theory (COLT95)*, pages 402–411, Santa Cruz, 1995 (ACM, New-York).

[OH95]       M. Opper and D. Haussler supervised learning, *Phys. Rev. Lett.*, 75:3772-3775, 1995.

[NP94a]      J.-P. Nadal and N. Parga. unsupervised learning. *Neural Computation*, 6:489–506, 1994.

[OK95]       M. Opper and W. Kinzel. In E. Domany J.L. van Hemmen and K. Schulten, editors, *Physics of Neural Networks*, pages 151–. Springer, 1995.

[Ris]        J. Rissanen. *IEEE Trans. on Information Theory*, 42 (1):40-47, 1996.

[RVdB96]     P. Reimann and C. Van den Broeck. *Phys. Rev. E*, 53 (4):3989–3998, 1996.

[VdB98]      C. Van den Broeck. In *proceedings of the TANC workshop* (Hong-Kong May 26-28, 1997).

[WN94]       T. Watkin and J.-P. Nadal. *J. Phys. A: Math. and Gen.*, 27:1899–1915, 1994.

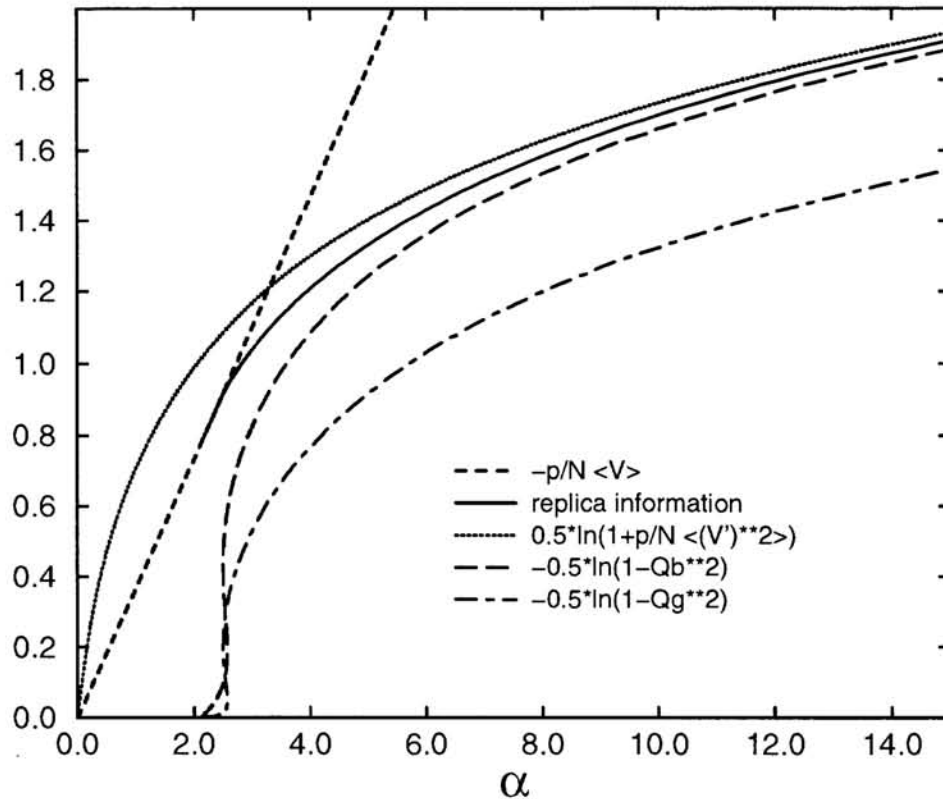

Figure 1: Dashed line is the linear bound on the mutual information $I(D|\theta)/N$. The latter, calculated with the replica technique, saturates the bound for $\alpha \leq \alpha_1$, and is the (lower) solid line for $\alpha > \alpha_1$. The special structure on fig.2 is not visible here due to the graph scale. The curve $-\frac{1}{2}ln(1 - Q_g{}^2)$ is a lower bound on the mutual information between the *Gibbs* estimator and $\theta$ (which would be equal to this bound if the conditional probability distribution of the estimator were Gaussian with mean $Q_g\theta$ and variance $1 - Q_g{}^2$). Shown also is the analogous curve $-\frac{1}{2}ln(1 - Q_b{}^2)$ for the *Bayes* estimator. In the limit $\alpha \to \infty$ these two latter Gaussian curves and the replica information $I(D|\theta)$, all converge toward the exact asymptotic behaviour, which can be expressed as $\frac{1}{2}ln(1 + \alpha < (\frac{dV(\lambda)}{d\lambda})^2 >)$ (upper solid line). This latter expression is, for any $p$, an upper bound for the two Gaussian curves.

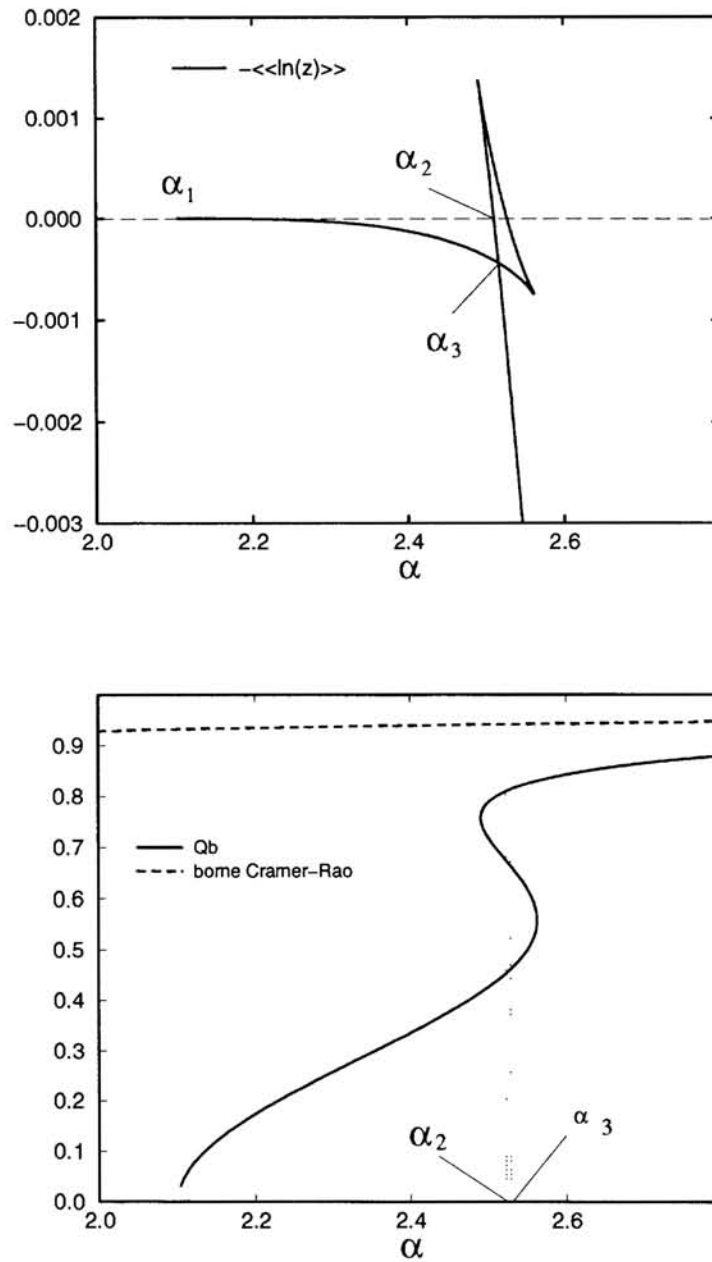

Figure 2: In the lower figure, the optimal learning curve $Q_b(\alpha)$ for $\rho = 1.2$ and $\sigma = 0.5$, as computed in (Buhot and Gordon, 1998) under the replica symetric ansatz. We have put the Cramer-Rao bound for this quantity. In the upper figure, the average free energy $- \ll lnZ \gg /N$. All the part above zero has to be rejected.
$\alpha_1 = 2.10$, $\alpha_2 = 2.515$ and $\alpha_3 = 2.527$